# Inference, Attention, and Decision in a Bayesian Neural Architecture

**Angela J. Yu**     **Peter Dayan**
Gatsby Computational Neuroscience Unit, UCL
17 Queen Square, London WC1N 3AR, United Kingdom.
feraina@gatsby.ucl.ac.uk     dayan@gatsby.ucl.ac.uk

## Abstract

We study the synthesis of neural coding, selective attention and perceptual decision making. A hierarchical neural architecture is proposed, which implements Bayesian integration of noisy sensory input and top-down attentional priors, leading to sound perceptual discrimination. The model offers an explicit explanation for the experimentally observed modulation that prior information in one stimulus feature (location) can have on an independent feature (orientation). The network's intermediate levels of representation instantiate known physiological properties of visual cortical neurons. The model also illustrates a possible reconciliation of cortical and neuromodulatory representations of uncertainty.

## 1 Introduction

A constant stream of noisy and ambiguous sensory inputs bombards our brains, informing on-going inferential processes and directing perceptual decision-making. Neurophysiologists and psychologists have long studied inference and decision-making in isolation, as well as the careful attentional filtering that is necessary to optimize them. The recent focus on their interactions poses an important opportunity and challenge for computational models. In this paper, we study an attentional task which involves all three components, and thereby directly confront their interaction. We first discuss the background of the individual elements; then describe our model.

The first element involves the representation and manipulation of uncertainty in sensory inputs and contextual information. There are two broad families of suggestions. One is microscopic, for which individual cortical neurons and populations either implicitly or explicitly represent the uncertainty. This spans a broad spectrum, from distributional codes that can also encode restricted aspects of uncertainty [1] to more exotic interpretations of codes as representing complex distributions [1, 2, 3, 4, 5]. The other family is macroscopic, with cholinergic (ACh) and noradrenergic (NE) neuromodulatory systems reporting computationally distinct forms of uncertainty to influence the way that information in differentially reliable cortical areas is integrated and learned [6, 7]. How microscopic and macroscopic families work together is hitherto largely unexplored.

The second element is selective attention and top-down influences over sensory processing. Here, the key challenge is to couple the many ideas about the way that attention *should*, from a sound statistical viewpoint, modify sensory processing, to the measurable effects of attention on the neural substrate. For instance, one typical consequence of (visual) featural and spatial attention is an increase in the activities of neurons in cortical populations repre-

senting those features, which is equivalent to *multiplying* their tuning functions by a factor [8]. Under the sort of probabilistic representational scheme in which the population activity codes for uncertainty in the underlying variable, it is of obvious importance to understand how this multiplication changes the implied uncertainty, and what statistical characteristic of the attention licenses this change [9].

The third element is the coupling between sensory processing and perceptual decisions. Implementational and computational issues underlying binary decisions, especially in simple cases, have been extensively explored, with psychologists [11, 12], and neuroscientists [13, 14] converging on common statistical [10] ideas about drift-diffusion processes.

In order to explore the interaction of these elements, we model an extensively studied attentional task (due to Posner [15]), in which probabilistic spatial cueing is used to manipulate attentional modulation of visual discrimination. We employ a hierarchical neural architecture in which top-down attentional priors are integrated with sequentially sampled sensory input in a sound Bayesian manner, using a logarithmic mapping between cortical neural activities and uncertainty [4]. In the model, the information provided by the cue is realized as a change in the prior distribution over the cued dimension (space). The effect of the prior is to eliminate inputs from spatial locations considered irrelevant for the task, thus improving discrimination in another dimension (orientation).

In section 2, we introduce the Posner task and give a Bayesian description of the computations underlying successful performance. In section 3, we describe the probabilistic semantics of the layers, and their functional connections, in the hierarchical neural architecture. In section 4, we compare the perceptual performance of the network to psychophysics data, and the intermediate layers' activities to the relevant physiological data.

## 2  Spatial Attention as Prior Information

In the classic version of Posner's task [15], a subject is presented with a cue that predicts the location of a subsequent target with a certain probability termed its *validity*. The cue is *valid* if it makes a correct prediction, and *invalid* otherwise. Subjects typically perform detection or discrimination on the target more rapidly and accurately on a valid-cue trial than an invalid one, reflecting cue-induced attentional modulation of visual processing and/or decision making [15]. This difference in reaction time or accuracy is often termed the validity effect [16], and depends on the cue validity [17].

We consider sensory stimuli with two feature dimensions, a periodic variable, orientation, $\phi = \phi^*$, about which decisions are to be made, and a linear variable, space, $\mu = \mu^*$ which is cued. The cue induces a top-down spatial prior, which we model as a mixture of a component sharply peaked at the cued location and a broader component capturing contextual and bottom-up saliency factors (including the possibility of invalidity). For simplicity, we use a Gaussian for the peaked component, and a uniform distribution for the broader one, although more complex priors of a similar nature would not change the model behavior: $p(\mu) = \gamma \mathcal{N}(\hat{\mu}, \nu^2) + (1-\gamma)c$. Given lower-layer activation patterns $\mathbf{X}_t \equiv \{\mathbf{x}_1, ..., \mathbf{x}_t\}$, assumed to be iid samples (with Gaussian noise) of bell-shaped tuning responses to the true underlying stimulus values $\mu^*, \phi^*$: $f_{ij}(\mu^*, \phi^*) = Z \exp(-(\mu_i - \mu^*)^2 / 2\sigma_\mu^2 + k \cos(\phi_j - \phi^*))$, the task is to infer a posterior distribution $P(\phi | \mathbf{X}_t)$, involving the following steps:

$$p(\mathbf{x}_t | \mu, \phi) = \prod_{ij} p(x_{ij}(t) | \mu, \phi) \qquad \text{Likelihood}$$

$$p(\phi | \mathbf{x}_t) = \int p(\mu, \phi) p(\mathbf{x}_t | \mu, \phi) d\mu \qquad \text{Prior-weighted marginalization}$$

$$p(\phi | \mathbf{X}_t) \propto p(\phi | \mathbf{x}_1^{t-1}) p(\mu, \phi | \mathbf{x}_t) \qquad \text{Temporal accumulation}$$

Because the marginalization step is weighted by the priors, a valid cue results in the inte-

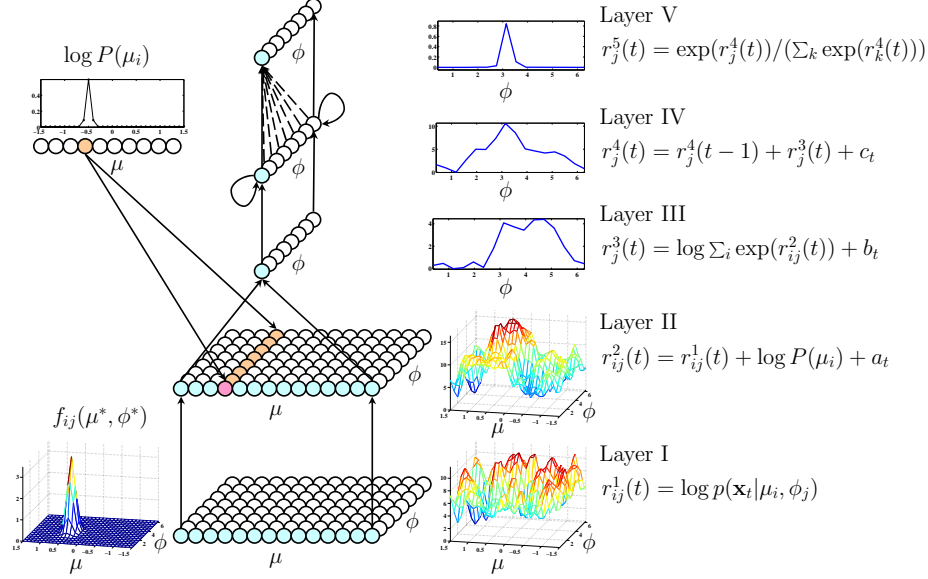

**Figure 1**: A Bayesian neural architecture. Layer I activities represent the log likelihood of the data given each possible setting of $\mu_i$ and $\phi_j$. This gives a noisy version of the smooth bell-shaped tuning curve (shown on the left). In layer II, the log likelihood of each $\mu_i$ and $\phi_j$ is modulated by the prior information $\log P(\mu_j)$, shown on the upper left. The prior in $\mu$ strongly suppresses the noisy input in the irrelevant part of the $\mu$ dimension, thus enabling improved inference based on the underlying tuning response $f_{ij}$. The layer III neurons represent the log marginal posterior of $\phi$ by integrating out the $\mu$ dimension of layer II activities. Layer IV neurons combine recurrent information and feedforward input from layer III to compute the log marginal posterior given all data so far observed. Layer V computes the cumulative posterior distribution of $\phi$ using a softmax operation. Due to the strong nonlinearity of softmax, its activity is much more peaked than in layer III and IV. Solid lines in the diagram represent excitatory connections, dashed lines inhibitory. Blue circles illustrate how the activities of one row of inputs in Layer I travels through the hierarchy to affect the final decision layer. Brown circles illustrate how one unit in the spatial prior layer comes into the integration process.

gration of more "signal" and less "noise" into the marginal posterior, whereas the opposite results from an invalid cue. To turn this on-line posterior into a decision $\hat{\phi}$, we use an extension of the Sequential Probability Ratio Test (SPRT [10]): observe $\mathbf{x}_1$, $\mathbf{x}_2$, ... until the first time that $\max P(\phi_j|\mathbf{X}_t)$ exceeds a fixed threshold $q$, then terminate the observation process and report $\hat{\phi} = \mathrm{argmax} P(\phi_j|\mathbf{X}_t)$ as the estimate of $\phi$ for the current trial.

## 3 A Bayesian Neural Architecture

The neural architecture implements the above computational steps exactly through a logarithmic transform, and has five layers (Fig 1). In layer I, activity of neuron $ij$, $r_{ij}^1(t)$, reports the log likelihood, $\log p(\mathbf{x}_t|\mu_i, \phi_j)$ (throughout, we discretize space and orientation). Layer II combines this log likelihood information with the prior, $r_{ij}^2(t) = r_{ij}^1(t) + \log P(\mu_i) + a_t$, to yield the joint log posterior up to an additive constant $a_t$ that makes $\min r_{ij}^2 = 0$. Layer III performs the marginalization $r_j^3(t) = \log \sum_i \exp(r_{ij}^2) + b_t$, to give the marginal posterior in $\phi$ (up to a constant $b_t$ that makes $\min r_j^3(t) = 0$). While this step ('log-of-sums') looks computationally formidable for neural hardware, it has been shown [4] that under certain conditions it can be well approximated by a (weighted) 'sum-of-logs' $r_j^3(t) \approx \sum_i c_i r_{ij}^2 + b_t$, where $c_i$ are weights optimized to minimize approximation error. Layer IV neurons combine recurrent information and feedforward input from layer III to compute the log marginal

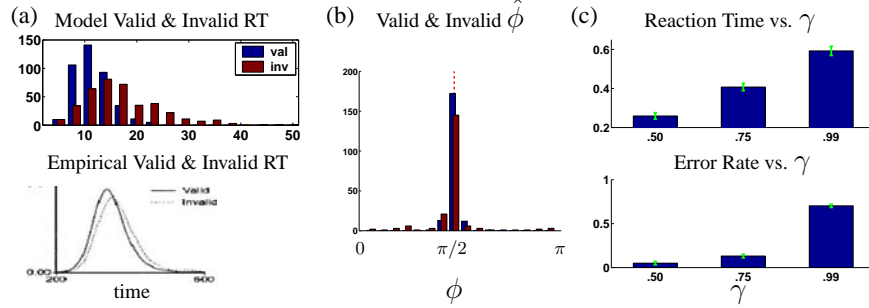

**Figure 2**: Validity effect and dependence on $\gamma$. (a) The distribution of reaction times for the invalid condition ($\gamma = 0.5$) has a greater mean and longer tail than the valid condition in model simulation results (top). Compare to similar results (bottom) from a Posner task in rats [18]. (b) Distribution of inferred $\hat{\phi}$ is more tightly clustered around the true $\phi^*$ (red dashed line) in valid case (blue) than the invalid case (red). $\gamma = 0.75$ (c) Validity effect, in both reaction time (top) and error rate (bottom) increases with increasing $\gamma$. $\{\mu_i\} = \{-1.5, -1.4, ..., 1.5\}$, $\{\phi_j\} = \{\pi/8, 2\pi/8, ..., 16\pi/8\}$, $\sigma_\mu = 0.1$, $\sigma_\phi = \pi/16$, $q = 0.90$, $\mu^* = 0.5$, $\gamma \in \{0.5, .75, .99\}$, $\nu = 0.05$, 300 trials each of valid and invalid trials. 100 trials of each $\gamma$ value.

posterior given all data so far observed, $r_j^4(t) = r_j^4(t-1) + r_j^3(t) + c_t$, up to a constant $c_t$. Finally, layer V neurons perform a softmax operation to retrieve the exact marginal posterior, $r_j^5(t) = \exp(r_j^4)/\sum_k \exp(r_k^4) = P(\phi_j | \mathbf{X}_t)$, with the additive constants dropping out. Note that a pathway parallel to III-IV-V consisting of neurons that only care about $\mu$ and not $\phi$ can be constructed in exactly the same manner. Its corresponding layers would report $\log p(\mathbf{x}_t, \mu_i)$, $\log p(\mathbf{X}_t, \mu_i)$, and $p(\mu_i | \mathbf{X}_t)$. An example of activities at each layer of the network, along with the choice of prior $p(\mu)$ and tuning function $f_{ij}$, is shown in Fig 1.

## 4    Results

We first verify that the model indeed exhibits the cue-induced validity effect, *ie* shorter RT and greater accuracy for valid-cue trials than invalid ones. "Reaction time" on a trial is the number of iid samples necessary to reach a decision, and "error rate" is the average angular distance between the estimated $\hat{\phi}$ and the true $\phi^*$. Figure 2 shows simulation results for 300 trials each of valid and invalid cue trials, for different values of $\gamma$, reflecting the model's belief as to cue validity. Reassuringly, the RT distribution for valid-cue trials distribution is tighter and left-shifted compared to invalid-cue trials (Figure 2(a), top panel), as observed in experimental data [15, 18] (Fig 2(a), bottom panel); (b) shows that accuracy is also higher for valid-cue trials. Consistent with data from a human Posner task [17], (c) shows that the VE increases with increasing perceived cue validity, as parameterized by $\gamma$, in both reaction times and error rates (precluding a simple speed-error trade-off).

Since we have an explicit model of not only the "behavioral output" but also the whole neural hierarchy, we can relate activities at various levels of representation to existing physiological data. Ample evidence indicates that spatial attention to one side of the visual field increases stimulus-induced activities in the corresponding part of the visual cortex [19, 20]. Fig 3(a) shows that our model qualitatively reproduces this effect; indeed it increases with $\gamma$, the perceived cue validity. Electrophysiological data also shows that spatial attention has a multiplicative effect on orientation tuning responses in visual cortical neurons [8] (Fig 3(b)). We see a similar phenomenon in the layer IV neurons (Fig 3(c); layer III similar, data not shown). Fig 3(d) is a scatter-plot of $\langle \log p(\mathbf{x}_t, \phi_j) + c_1 \rangle_t$ for the valid condition versus the invalid condition, for various values of $\gamma$, along with the slope fit to the experiment of Fig 3(b) (Layer III similar, data not shown). The linear least square error fits are good, and the slope increases with increasing confidence in the cued location (larger $\gamma$). In

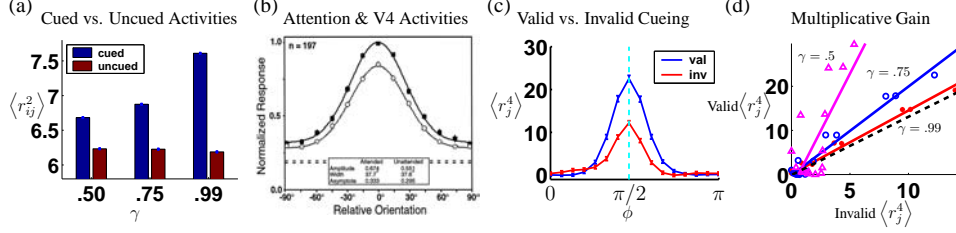

**Figure 3**: Multiplicative gain modulation by spatial attention. (a) $r_{ij}^2$ activities, averaged over the half of layer II where the prior peaks, are greater for valid (blue, left) than invalid (red, right) conditions. (b) Experimentally observed multiplicative modulation of V4 orientation tunings by spatial attention [8]. (c) Similar multiplicative effect in layer IV in the model. (d) Linear fits to scatter-plot of layer III activities for valid cue condition vs. invalid cue condition show that the slope is greatest for large $\gamma$ and smallest for small $\gamma$ (magenta: $\gamma = 0.99$, blue: $\gamma = 0.75$, red: $\gamma = 0.5$, black: linear fit to study in (b)). Simulation parameters are same as in Fig 2. Error bars: standard errors of the mean.

the model, the slope not only depends on $\gamma$ but also the noise model, the discretization, and so on, so the comparison of Figure 3(d) should be interpreted loosely.

In valid cases, the effect of attention is to increase the certainty in the posterior marginal over $\phi$, since the correct prior allows the relative suppression of noisy input from the irrelevant part of space. Were the posterior marginal exactly Gaussian, the increased certainty would translate into a decreased variance. For Gaussian probability distributions, logarithmic coding amounts to something close to a quadratic (adjusted for the circularity of orientation), with a curvature determined by the variance. Decreasing the variance increases the curvature, and therefore has a multiplicative effect on the activities (as in figure 3).

The approximate gaussianity of the marginal posterior comes from the accumulation of many independent samples over time and space, and something like the central limit theorem. While it is difficult to show this multiplicative modulation rigorously, we can at least demonstrate it mathematically for the case where the spatial prior is very sharply peaked at its Gaussian mean $\hat{y}$. In this case, $(\langle \log p_1(\mathbf{x}(t), \phi_j)\rangle_t + c_1)/(\langle \log p_2(\mathbf{x}(t), \phi_j)\rangle_t + c_2) \approx R$, where $c_1$, $c_2$, and R are constants independent of $\phi_j$ and $\mu_i$. Based on the peaked prior assumption, $p(\mu) \approx \delta(\mu - \hat{\mu})$, we have $p(\mathbf{x}(t), \phi) = \int p(\mathbf{x}(t)|\mu, \phi)p(\mu)p(\phi) \approx p(\mathbf{x}(t)|\phi, \hat{\mu})$. We can expand $\log p(\mathbf{x}(t)|\hat{\mu}, \phi)$ and compute its average over time

$$\langle \log p(\mathbf{x}(t)|\hat{\mu}, \phi)\rangle_t = C - \frac{N}{2\sigma_n^2}\left\langle (f_{ij}(\mu^*, \phi^*) - f_{ij}(\hat{\mu}, \phi))^2 \right\rangle_{ij}. \tag{1}$$

Then using the tuning function defined earlier, we can compare the joint probabilities given valid (val) and invalid (inv) cues:

$$\frac{\langle \log p_{\text{val}}(\mathbf{x}(t), \phi)\rangle_t}{\langle \log p_{\text{inv}}(\mathbf{x}(t), \phi)\rangle_t} = \frac{\alpha_1 - \beta \left\langle e^{-(\mu_i - \mu^*)^2/\sigma_\mu^2}\right\rangle_i \langle g(\phi)\rangle_j}{\alpha_2 - \beta \left\langle e^{-((\mu_i - \mu^*)^2 + (\mu_i - \hat{\mu})^2)/2\sigma_\mu^2}\right\rangle_i \langle g(\phi)\rangle_j}, \tag{2}$$

and therefore, 
$$\frac{\langle \log p_{val}(\mathbf{x}_t, \phi)\rangle_t + c_1}{\langle \log p_{inv}(\mathbf{x}_t, \phi)\rangle_t + c_2} \approx e^{(\mu^* - \hat{\mu})^2/(4\sigma_\mu^2)} = R. \tag{3}$$

The derivation for a multiplicative effect on layer IV activities is very similar.

Another aspect of intermediate representation of interest is the way attention modifies the evidence accumulation process over time. Fig 4 show the effect of cueing on the activities of neuron $r_{j*}^5(t)$, or $P(\phi^*|\mathbf{X}_t)$, for all trials with correct responses. The mean activity trajectory is higher for the valid cue case than the invalid one: in this case, spatial attention mainly acts through increasing the rate of evidence accumulation after stimulus onset

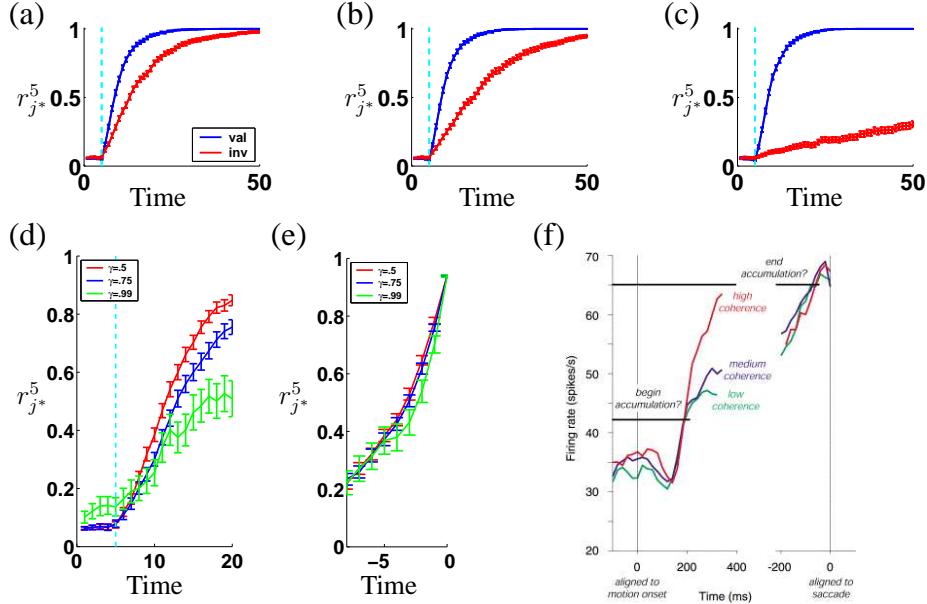

**Figure 4**: Accumulation of iid samples in orientation discrimination, and dependence on prior belief about stimulus location. (a-c) Average activity of neuron $r_{j*}^5$, which represents $P(\phi^*|\mathbf{X}_t)$, saturates to $100\%$ certainty much faster for valid cue trials (blue) than invalid cue trials (red). The difference is more drastic when $\gamma$ is larger, or when there is more prior confidence in the cued target location. (a) $\gamma = 0.5$, (b) $\gamma = 0.75$, (c) $\gamma = 0.99$. Cyan dashed line indicates stimulus onset. (d) First 15 time steps (from stimulus onset) of the invalid cue traces from (a-c) are aligned to stimulus onset; cyan line denotes stimulus onset. The differential rates of rise are apparent. (e) Last 8 time steps of the invalid traces from (a-c) are aligned to decision threshold-crossing; there is no clear separation as a function $\gamma$. (f) Multiplicative gain modulation of attention on V4 orientation tuning curves. Simulation parameters are same as in Fig 2.

(steeper rise). This attentional effect is more pronounced when the system is more confident about its prior information ((a) $\gamma = 0.5$, (b) $\gamma = 0.75$, (c) $\gamma = 0.99$). Effectively, increasing $\gamma$ for invalid-cue trials is increasing input *noise*. Figure 4 (d) shows the average traces for invalid-cueing trials aligned to the stimulus onset and (e) to the decision threshold crossing. These results bear remarkable similarities to the LIP neuronal activities recorded during monkey perceptual decision-making [13] (shown in (f)). In the stimulus-aligned case, the traces rise linearly at first and then tail off somewhat, and the rate of rise increases for lower (effective) noise. In the decision-aligned case, the traces rise steeply and together. All these characteristics can also be seen in the experimental results in (f), where the input noise level is explicitly varied.

## 5 Discussion

We have presented a hierarchical neural architecture that implements optimal probabilistic integration of top-down information and sequentially observed data. We consider a class of attentional tasks for which top-down modulation of sensory processing can be conceptualized as changes in the prior distribution over implicit stimulus dimensions. We use the specific example of the Posner spatial cueing task to relate the characteristics of this neural architecture to experimental literature. The network produces a reaction time distribution and error rates that qualitatively replicate experimental data. The way these measures depend on valid versus invalid cueing, and on the exact perceived validity of the cue, are similar to those observed in attentional experiments. Moreover, the activities in various

levels of the hierarchy resemble electrophysiologically recorded activities in the visual cortical neurons during attentional modulation and perceptual discrimination, lending farther credence to the particular encoding and computational mechanisms that we have proposed. In particular, the intermediate layers demonstrate a multiplicative gain modulation by attention, as observed in primate V4 neurons [8]; and the temporal behavior of the final layer, representing the marginal posterior, qualitative replicates the experimental observation that LIP neurons show noise-dependent firing rate increase when aligned to stimulus onset, and noise-independent rise when aligned to the decision [13].

Our results illustrate the important concept that priors in a variable in one dimension (space) can dramatically alter the inferential performance in a completely independent variable dimension (orientation). In this case, the spatial prior affects the marginal posterior over $\phi$ by altering the relative importance of joint posterior terms in the marginalization process. This leads to the difference in performance between valid and invalid trials, a difference that increases with $\gamma$. This model elaborates on an earlier phenomenological model [9], by showing explicitly how marginalizing (in layer III) over activities biased by the prior (in layer II) produces the effect.

This work has various theoretical and experimental implications. The model presents one possible reconciliation of cortical and neuromodulatory representations of uncertainty. The sensory-driven activities (layer I in this model) themselves encode bottom-up uncertainty, including sensory receptor noise and any processing noise that have occurred up until then. The top-down information, which specifies the Gaussian component of the spatial prior $p(\mu)$, involves two kinds of uncertainty. One determines the locus and spatial extent of visual attention, the other specifies the relative importance of this top-down bias compared to the bottom-up stimulus-driven input. The first is highly specific in modality and featural dimension, presumably originating from higher visual cortical areas (*eg* parietal cortex for spatial attention, inferotemporal cortex for complex featural attention). The second is more generic and may affect different featural dimensions and maybe even different modalities simultaneously, and is thus more appropriately signalled by a diffusely-projecting neuromodulator such as ACh. This characterization is also in keeping with our previous models of ACh [21, 7] and experimental data showing that ACh selectively suppresses cortico-cortical transmission relative to bottom-up processing in primary sensory cortices [22].

The perceptual decision strategy employed in this model is a natural multi-dimensional extension of SPRT [10], by monitoring the first-time passage of any *one* of the posterior values crossing a fixed decision threshold.. Note that the distribution of reaction times is skewed to the right (Fig 2(a)), as is commonly observed in visual discrimination tasks [11]. For *binary* decision tasks modeled using continuous diffusion processes [10, 11, 12, 13, 14], this skew arises from the properties of the first-passage time distribution (the time at which a diffusion barrier is first breached, corresponding to a fixed threshold confidence level in the binary choice). Our multi-choice decision-making realization of visual discrimination, as an extension of SPRT, also retains this skewed first-passage time distribution. Given that SPRT is optimal for binary decisions (smallest average response time for a given error rate), and that MAP estimate is optimal for 0-1 loss, we conjecture that our particular n-dim generalization of SPRT should be optimal for sequential decision-making under 0-1 loss. This is an area of active research.

There are several important open issues. One is that of noise: our network performs exact Bayesian inference when activities are deterministic. The potentially deleterious effects of noise, particularly in log probability space, needs to be explored. Another important question is how uncertainty in signal strength, including the absence of a signal, can be detected and encoded. If the stimulus strength is unknown and can vary over time, then naive integration of bottom-up inputs ignoring the signal-to-noise ratio is no longer optimal. Based on a slightly different task involving sustained attention or vigilance [23], Brown *et al* [24] have made the interesting suggestion that one role for noradrenergic neuromodulation is

to implement a change in the integration strategy when a stimulus is detected. We have also addressed this issue by ascribing to phasic norepinephrine a related but distinct role in signaling unexpected state uncertainty (in preparation).

## Acknowledgement

We are grateful to Eric Brown, Jonathan Cohen, Phil Holmes, Peter Latham, and Iain Murray for helpful discussions. Funding was from the Gatsby Charitable Foundation.

## References

[1] Zemel, R S, Dayan, P, & Pouget, A (1998). Probabilistic interpretation of population codes. *Neural Comput* **10**: 403-30.
[2] Sahani, M & Dayan, P (2003). Doubly distributional population codes: simultaneous representation of uncertainty and multiplicity. *Neural Comput* **15**: 2255-79.
[3] Barber, M J, Clark, J W, & Anderson, C H (2003). Neural representation of probabilistic information. *Neural Comput* **15**: 1843-64
[4] Rao, R P (2004). Bayesian computation in recurrent neural circuits. *Neural Comput* **16**: 1-38.
[5] Weiss, Y & Fleet D J(2002). Velocity likelihoods in biological and machine vision. In *Prob Models of the Brain: Perc and Neural Function*. Cambridge, MA: MIT Press.
[6] Dayan, P & Yu, A J (2002). Acetylcholine, uncertainty, and cortical inference. In *Adv in Neural Info Process Systems 14*.
[7] Yu, A J & Dayan, P (2003). Expected and unexpected uncertainty: ACh and NE in the neocortex. In *Adv in Neural Info Process Systems 15*.
[8] McAdams, C J & Maunsell, J H R (1999). Effects of attention on orientation-tuning functions of single neurons in Macaque cortical area V4. *J. Neurosci* **19**: 431-41.
[9] Dayan, P & Zemel R S (1999). Statistical models and sensory attention. In *ICANN 1999*.
[10] Wald, A (1947). *Sequential Analysis*. New York: John Wiley & Sons, Inc.
[11] Luce, R D (1986). *Response Times: Their Role in Inferring Elementary Mental Organization*. New York: Oxford Univ. Press.
[12] Ratcliff, R (2001). Putting noise into neurophysiological models of simple decision making. *Nat Neurosci* **4**: 336-7.
[13] Gold, J I & Shadlen, M N (2002). Banburismus and the brain: decoding the relationship between sensory stimuli, decisions, and reward. *Neuron* **36**: 299-308.
[14] Bogacz, Brown, Moehlis, Holmes, & Cohen (2004). The physics of optimal decision making: a formal analysis of models of performance in two-alternative forced choice tasks, in press.
[15] Posner, M I (1980). Orienting of attention. *Q J Exp Psychol* **32**: 3-25.
[16] Phillips, J M, et al (2000). Cholinergic neurotransmission influences overt orientation of visuospatial attention in the rat. *Psychopharm* **150**:112-6.
[17] Yu, A J *et al* (2004). Expected and unexpected uncertainties control allocation of attention in a novel attentional learning task. *Soc Neurosci Abst* 30:176.17.
[18] Bowman, E M, Brown, V, Kertzman, C, Schwarz, U, & Robinson, D L (2003). Covert orienting of attention in Macaques: I. effects of behavioral context. *J Neurophys* **70**: 431-434.
[19] Reynolds, J H & Chelazzi, L (2004). Attentional modulation of visual processing. *Annu Rev Neurosci* **27**: 611-47.
[20] Kastner, S & Ungerleider, L G (2000). Mechanisms of visual attention in the human cortex. *Annu Rev Neurosci* **23**: 315-41.
[21] Yu, A J & Dayan, P (2002). Acetylcholine in cortical inference. *Neural Networks* **15**: 719-30.
[22] Kimura, F, Fukuada, M, & Tusomoto, T (1999). Acetylcholine suppresses the spread of excitation in the visual cortex revealed by optical recording: possible differential effect depending on the source of input. *Eur J Neurosci* **11**: 3597-609.
[23] Rajkowski, J, Kubiak, P, & Aston-Jones, P (1994). Locus coeruleus activity in monkey: phasic and tonic changes are associated with altered vigilance. *Synapse* **4**: 162-4.
[24] Brown, E *et al* (2004). Simple neural networks that optimize decisions. *Int J Bifurcation and Chaos*, in press.
